# Stacked Density Estimation

**Padhraic Smyth** *
Information and Computer Science
University of California, Irvine
CA 92697-3425
smyth@ics.uci.edu

**David Wolpert**
NASA Ames Research Center
Caelum Research
MS 269-2, Mountain View, CA 94035
dhw@ptolemy.arc.nasa.gov

## Abstract

In this paper, the technique of stacking, previously only used for supervised learning, is applied to unsupervised learning. Specifically, it is used for non-parametric multivariate density estimation, to combine finite mixture model and kernel density estimators. Experimental results on both simulated data and real world data sets clearly demonstrate that stacked density estimation outperforms other strategies such as choosing the single best model based on cross-validation, combining with uniform weights, and even the single best model chosen by "cheating" by looking at the data used for independent testing.

## 1 Introduction

Multivariate probability density estimation is a fundamental problem in exploratory data analysis, statistical pattern recognition and machine learning. One frequently estimates density functions for which there is little prior knowledge on the shape of the density and for which one wants a flexible and robust estimator (allowing multimodality if it exists). In this context, the methods of choice tend to be finite mixture models and kernel density estimation methods. For mixture modeling, mixtures of Gaussian components are frequently assumed and model choice reduces to the problem of choosing the number $k$ of Gaussian components in the model (Titterington, Smith and Makov, 1986) . For kernel density estimation, kernel shapes are typically chosen from a selection of simple unimodal densities such as Gaussian, triangular, or Cauchy densities, and kernel bandwidths are selected in a data-driven manner (Silverman 1986; Scott 1994).

As argued by Draper (1996), model uncertainty can contribute significantly to pre-

dictive error in estimation. While usually considered in the context of supervised learning, model uncertainty is also important in unsupervised learning applications such as density estimation. Even when the model class under consideration contains the true density, if we are only given a finite data set, then there is always a chance of selecting the wrong model. Moreover, even if the correct model is selected, there will typically be estimation error in the parameters of that model. These difficulties are summarized by writing

$$P(f \mid D) = \sum_M \int d\theta_M P(\theta_M \mid D, M) \times P(M \mid D) \times f_{M, \theta_M}, \qquad (1)$$

where $f$ is a density, $D$ is the data set, $M$ is a model, and $\theta_M$ is a set of values for the parameters for model $M$. The posterior probability $P(M \mid D)$ reflects model uncertainty, and the posterior $P(\theta_M \mid D, M)$ reflects uncertainty in setting the parameters even once one knows the model. Note that if one is privy to $P(M, \theta_M)$, then Bayes' theorem allows us to write out both of our posteriors explicitly, so that we explicitly have $P(f \mid D)$ (and therefore the Bayes-optimal density) given by a weighted average of the $f_{M, \theta_M}$. (See also Escobar and West (1995)). However even when we know $P(M, \theta_M)$, calculating the combining weights can be difficult. Thus, various approximations and sampling techniques are often used, a process that necessarily introduces extra error (Chickering and Heckerman 1997). More generally, consider the case of mis-specified models where the model class does not include the true model, so our presumption for $P(M, \theta_M)$ is erroneous. In this case often one should again average.

Thus, a natural approach to improving density estimators is to consider empirically-driven combinations of multiple density models. There are several ways to do this, especially if one exploits previous combining work in supervised learning. For example, Ormontreit and Tresp (1996) have shown that "bagging" (uniformly weighting different parametrizations of the same model trained on different bootstrap samples), originally introduced for supervised learning (Breiman 1996a), can improve accuracy for mixtures of Gaussians with a fixed number of components. Another supervised learning technique for combining different types of models is "stacking" (Wolpert 1992), which has been found to be very effective for both regression and classification (e.g., Breiman (1996b)). This paper applies stacking to density estimation, in particular to combinations involving kernel density estimators together with finite mixture model estimators.

## 2 Stacked Density Estimation

### 2.1 Background on Density Estimation with Mixtures and Kernels

Consider a set of $d$ real-valued random variables $\underline{X} = \{X^1, \ldots, X^d\}$ Upper case symbols denote variable names (such as $X^j$) and lower-case symbols a particular value of a variable (such as $x^j$). $\underline{x}$ is a realization of the vector variable $\underline{X}$. $f(\underline{x})$ is shorthand for $f(\underline{X} = \underline{x})$ and represents the joint probability distribution of $\underline{X}$. $D = \{\underline{x}_1, \ldots, \underline{x}_N\}$ is a training data set where each sample $\underline{x}_i, 1 \leq i \leq N$ is an independently drawn sample from the underlying density function $\bar{f}(\underline{x})$.

A commonly used model for density estimation is the *finite mixture model* with $k$ components, defined as:

$$f^k(\underline{x}) = \sum_{j=1}^k \alpha_j g_j(\underline{x}), \qquad (2)$$

where $\sum_{j=1}^{k} \alpha_j = 1$. The component $g_j$'s are usually relatively simple unimodal densities such as Gaussians. Density estimation with mixtures involves finding the locations, shapes, and weights of the component densities from the data (using for example the Expectation-Maximization (EM) procedure). *Kernel density estimation* can be viewed as a special case of mixture modeling where a component is centered at each data point, given a weight of $1/N$, and a common covariance structure (kernel shape) is estimated from the data.

The quality of a particular probabilistic model can be evaluated by an appropriate scoring rule on independent out-of-sample data, such as the test set log-likelihood (also referred to as the log-scoring rule in the Bayesian literature). Given a test data set $D^{test}$, the test log likelihood is defined as

$$\log f(D^{test}|f^k(\underline{x})) = \sum_{D^{test}} \log f^k(\underline{x}_i) \tag{3}$$

This quantity can play the role played by classification error in classification or squared error in regression. For example, cross-validated estimates of it can be used to find the best number of clusters to fit to a given data set (Smyth, 1996).

## 2.2 Background on Stacking

Stacking can be used either to combine models or to improve a single model. In the former guise it proceeds as follows. First, subsamples of the training set are formed. Next the models are all trained on one subsample and resultant joint predictive behavior on another subsample is observed, together with information concerning the optimal predictions on the elements in that other subsample. This is repeated for other pairs of subsamples of the training set. Then an additional ("stacked") model is trained to learn, from the subsample-based observations, the relationship between the observed joint predictive behavior of the models and the optimal predictions. Finally, this learned relationship is used in conjunction with the predictions of the individual models being combined (now trained on the entire data set) to determine the full system's predictions.

## 2.3 Applying Stacking to Density Estimation

Consider a set of $M$ different density models, $f_m(\underline{x}), 1 \leq m \leq M$. In this paper each of these models will be either a finite mixture with a fixed number of component densities or a kernel density estimate with a fixed kernel and a single fixed global bandwidth in each dimension. (In general though no such restrictions are needed.) The procedure for stacking the $M$ density models is as follows:

1. Partition the training data set $D$ $v$ times, exactly as in $v$-fold cross validation (we use $v = 10$ throughout this paper), and for each fold:

    (a) Fit each of the $M$ models to the training portion of the partition of $D$.

    (b) Evaluate the likelihood of each data point in the test partition of $D$, for each of the $M$ fitted models.

2. After doing this one has $M$ density estimates for each of $N$ data points, and therefore a matrix of size $N \times M$, where each entry is $f_m(\underline{x}_i)$, the out-of-sample likelihood of the $m$th model on the $i$th data point.

3. Use that matrix to estimate the combination coefficients $\{\beta_1, \ldots, \beta_M\}$ that maximize the log-likelihood at the points $\underline{x}_i$ of a stacked density model of

the form:

$$f_{\text{stacked}}(\underline{x}) = \sum_{m=1}^{M} \beta_m f_m(\underline{x}).$$

Since this is itself a mixture model, but where the $f_m(\underline{x}_i)$ are fixed, the EM algorithm can be used to (easily) estimate the $\beta_m$.

4. Finally, re-estimate the parameters of each of the $m$ component density models using *all* of the training data $D$. The stacked density model is then the linear combination of those density models, with combining coefficients given by the $\beta_m$.

## 3   Experimental Results

In our stacking experiments $M = 6$: three triangular kernels with bandwidths of $0.1, 0.4$, and $1.5$ of the standard deviation (of the full data set) in each dimension, and three Gaussian mixture models with $k = 2, 4$, and $8$ components. This set of models was chosen to provide a reasonably diverse representational basis for stacking. We follow roughly the same experimental procedure as described in Breiman (1996b) for stacked regression:

- Each data set is randomly split into training and test partitions 50 times, where the test partition is chosen to be large enough to provide reasonable estimates of out-of-sample log-likelihood.
- The following techniques are run on each training partition:
  1. **Stacking:** The stacked combination of the six constituent models.
  2. **Cross-Validation:** The single best model as indicated by the maximum likelihood score of the $M = 6$ single models in the $N \times M$ cross-validated table of likelihood scores.
  3. **Uniform Weighting:** A uniform average of the six models.
  4. **"Cheating:"** The best single model, i.e., the model having the largest likelihood on the *test* data partition,
  5. **Truth:** The true model structure, if the true model is one of the six generating the data (only valid for simulated data).
- The log-likelihoods of the models resulting from these techniques are calculated on the test data partition. The log-likelihood of a single Gaussian model (parameters determined on the training data) is subtracted from each model's log-likelihood to provide some normalization of scale.

### 3.1   Results on Real Data Sets

Four real data sets were chosen for experimental evaluation. The diabetes data set consists of 145 data points used in Gaussian clustering studies by Banfield and Raftery (1991) and others. Fisher's iris data set is a classic data set in 4 dimensions with 150 data points. Both of these data sets are thought to consist roughly of 3 clusters which can be reasonably approximated by 3 Gaussians. The Barney and Peterson vowel data (2 dimensions, 639 data points) contains 10 distinct vowel sounds and so is highly multi-modal. The star-galaxy data (7 dimensions, 499 data points) contains non-Gaussian looking structure in various 2d projections.

Table 1 summarizes the results. In all cases stacking had the highest average log-likelihood, even out-performing "cheating" (the single best model chosen from the test data). (Breiman (1996b) also found for regression that stacking outperformed

Table 1: Relative performance of stacking multiple mixture models, for various data sets, measured (relative to the performance of a single Gaussian model) by mean log-likelihood on test data partitions. The maximum for each data set is underlined.

| Data Set | Gaussian | Cross-Validation | "Cheating" | Uniform | Stacking |
|---|---|---|---|---|---|
| Diabetes | -352.9 | 27.8 | 30.4 | 29.2 | 31.8 |
| Fisher's Iris | -52.6 | 18.3 | 21.2 | 18.3 | 22.5 |
| Vowel | 128.9 | 53.5 | 54.6 | 40.2 | 55.8 |
| Star-Galaxy | -257.0 | 678.9 | 721.6 | 789.1 | 888.9 |

Table 2: Average across 20 runs of the stacked weights found for each constituent model. The columns with $h = \ldots$ are for the triangular kernels and the columns with $k = \ldots$ are for the Gaussian mixtures.

| Data Set | $h=0.1$ | $h=0.4$ | $h=1.5$ | $k=2$ | $k=4$ | $k=8$ |
|---|---|---|---|---|---|---|
| Diabetes | 0.01 | 0.09 | 0.03 | 0.13 | 0.41 | 0.32 |
| Fisher's Iris | 0.02 | 0.16 | 0.00 | 0.26 | 0.40 | 0.16 |
| Vowel | 0.00 | 0.25 | 0.00 | 0.02 | 0.20 | 0.53 |
| Star-Galaxy | 0.00 | 0.04 | 0.03 | 0.03 | 0.27 | 0.62 |

the "cheating" method.) We considered two null hypotheses: stacking has the same predictive accuracy as cross-validation, and it has the same accuracy as uniform weighting. Each hypothesis can be rejected with a chance of less than 0.01% of being incorrect, according to the Wilcoxon signed-rank test i.e., the observed differences in performance are extremely strong even given the fact that this particular test is not strictly applicable in this situation.

On the vowel data set uniform weighting performs much worse than the other methods: it is closer in performance to stacking on the other 3 data sets. On three of the data sets, using cross-validation to select a single model is the worst method. "Cheating" is second-best to stacking except on the star-galaxy data, where it is worse than uniform weighting also: this may be because the star-galaxy data probably induces the greatest degree of mis-specification relative to this 6-model class (based on visual inspection).

Table 2 shows the averages of the stacked weight vectors for each data set. The mixture components generally got higher weight than the triangular kernels. The vowel and star-galaxy data sets have more structure than can be represented by any of the component models and this is reflected in the fact that for each most weight is placed on the most complex mixture model with $k = 8$.

## 3.2 Results on Simulated Data with no Model Mis-Specification

We simulated data from a 2-dimensional 4-Gaussian mixture model with a reasonable degree of overlap (this is the data set used in Ripley (1994) with the class labels removed) and compared the same models and combining/selection schemes as before, except that "truth" is also included, i.e., the scheme which always selects the true model structure with $k = 4$ Gaussians. For each training sample size, 20 different training data sets were simulated, and the mean likelihood on an independent test data set of size 1000 was reported.

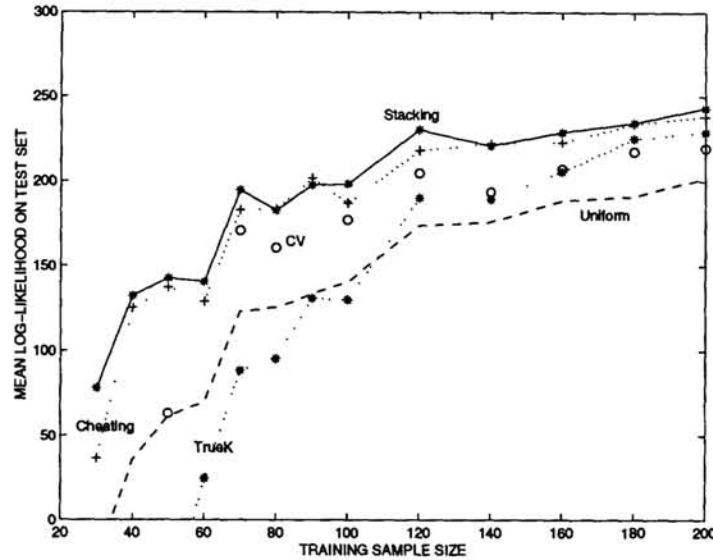

Figure 1: Plot of mean log-likelihood (relative to a single Gaussian model) for various density estimation schemes on data simulated from a 4-component Gaussian mixture.

Note that here we are assured of having the true model in the set of models being considered, something that is presumably never exactly the case in the real world (and presumably was not the case for the experiments recounted in Table 1.) Nonetheless, as indicated in (Figure 1), stacking performed about the same as the "cheating" method and significantly outperformed the other methods, including "truth." (Results where some of the methods had log-likelihoods *lower* than the single Gaussian are not shown for clarity).

The fact that "truth" performed poorly on the smaller sample sizes is due to the fact that with smaller sample sizes it was often better to fit a simpler model with reliable parameter estimates (which is what "cheating" typically would do) than a more complex model which may overfit (even when it is the true model structure). As the sample size increases, both "truth" and cross-validation approach the performance of "cheating" and stacking: uniform weighting is universally poorer as one would expect when the true model is within the model class. The stacked weights at the different sample sizes (not shown) start out with significant weight on the triangular kernel model and gradually shift to the $k = 2$ Gaussian mixture model and finally to the (true) $k = 4$ Gaussian model as sample size grows. Thus, stacking is seen to incur no penalty when the true model is within the model class being fit. In fact the opposite is true; for small sample sizes stacking outperforms other density estimation techniques which place full weight on a single (but poorly parametrized) model.

## 4   Discussion and Conclusions

Selecting a global bandwidth for kernel density estimation is still a topic of debate among statisticians. Stacking allows the possibility of side-stepping the issue of a single bandwidth by combining kernels with different bandwidths and different kernel shapes. A stacked combination of such kernel estimators is equivalent to using

a single composite kernel that is a convex combination of the underlying kernels. For example, kernel estimators based on finite support kernels can be regularized in a data-driven manner by combining them with infinite support kernels. The key point is that the shape and width of the resulting "effective" kernel is driven by the data.

It is also worth noting that by combining Gaussian mixture models with different $k$ values one gets a hierarchical "mixture of mixtures" model. This hierarchical model can provide a natural multi-scale representation of the data, which is clearly similar in spirit to wavelet density estimators, although the functional forms and estimation methodologies for each technique can be quite different. There is also a representational similarity to Jordan and Jacob's (1994) "mixture of experts" model where the weights are allowed to depend directly on the inputs. Exploiting that similarity, one direction for further work is to investigate adaptive weight parametrizations in the stacked density estimation context.

## Acknowledgements

The work of P.S. was supported in part by NSF Grant IRI-9703120 and in part by the Jet Propulsion Laboratory, California Institute of Technology, under a contract with the National Aeronautics and Space Administration.

## Footnotes

*Also with the Jet Propulsion Laboratory 525-3660, California Institute of Technology, Pasadena, CA 91109

## References

Banfield, J. D., and Raftery, A. E., 'Model-based Gaussian and non-Gaussian clustering,' *Biometrics*, 49, 803–821, 1993.

Breiman, L., 'Bagging predictors,' *Machine Learning*, 26(2), 123–140, 1996a.

Breiman, L., 'Stacked regressions,' *Machine Learning*, 24, 49–64, 1996b.

Chickering, D. M., and Heckerman, D., 'Efficient approximations for the marginal likelihood of Bayesian networks with hidden variables,' *Machine Learning*, in press.

Draper, D, 'Assessment and propagation of model uncertainty (with discussion),' *Journal of the Royal Statistical Society B*, 57, 45–97, 1995.

Escobar, M. D., and West, M., 'Bayesian density estimation and inference with mixtures,' *J. Am. Stat. Assoc.*, 90, 577-588, 1995.

Jordan, M. I. and Jacobs, R. A., 'Hierarchical mixtures of experts and the EM algorithm,' *Neural Computation*, 6, 181–214, 1994.

Madigan, D., and Raftery, A. E., 'Model selection and accounting for model uncertainty in graphical models using Occam's window,' *J. Am. Stat. Assoc.*, 89, 1535–1546, 1994.

Ormeneit, D., and Tresp, V., 'Improved Gaussian mixture density estimates using Bayesian penalty terms and network averaging,' in *Advances in Neural Information Processing 8*, 542–548, MIT Press, 1996.

Ripley, B. D. 1994. 'Neural networks and related methods for classification (with discussion),' *J. Roy. Stat. Soc. B*, 56, 409–456.

Smyth, P.,'Clustering using Monte-Carlo cross-validation,' in *Proceedings of the Second International Conference on Knowledge Discovery and Data Mining*, Menlo Park, CA: AAAI Press, pp.126–133, 1996.

Titterington, D. M., A. F. M. Smith, U. E. Makov, *Statistical Analysis of Finite Mixture Distributions*, Chichester, UK: John Wiley and Sons, 1985

Wolpert, D. 1992. 'Stacked generalization,' *Neural Networks*, 5, 241–259.